# Clustering Sequence Sets for Motif Discovery

**Jong Kyoung Kim  and  Seungjin Choi**
Department of Computer Science
Pohang University of Science and Technology
San 31 Hyoja-dong, Nam-gu
Pohang 790-784, Korea
{blkimjk,seungjin}@postech.ac.kr

## Abstract

Most of existing methods for DNA motif discovery consider only a single set of sequences to find an over-represented motif. In contrast, we consider multiple sets of sequences where we group sets associated with the same motif into a cluster, assuming that each set involves a single motif. Clustering sets of sequences yields clusters of coherent motifs, improving signal-to-noise ratio or enabling us to identify multiple motifs. We present a probabilistic model for DNA motif discovery where we identify multiple motifs through searching for patterns which are shared across multiple sets of sequences. Our model infers cluster-indicating latent variables and learns motifs simultaneously, where these two tasks interact with each other. We show that our model can handle various motif discovery problems, depending on how to construct multiple sets of sequences. Experiments on three different problems for discovering DNA motifs emphasize the useful behavior and confirm the substantial gains over existing methods where only a single set of sequences is considered.

## 1 Introduction

Discovering how DNA-binding proteins called *transcription factors* (TFs) regulate gene expression programs in living cells is fundamental to understanding transcriptional regulatory networks controlling development, cancer, and many human diseases. TFs that bind to specific cis-regulatory elements in DNA sequences are essential for mediating this transcriptional control. The first step toward deciphering this complex network is to identify functional binding sites of TFs referred to as *motifs*.

We address the problem of discovering sequence motifs that are enriched in a given target set of sequences, compared to a background model (or a set of background sequences). There have been extensive research works on statistical modeling of this problem (see [1] for review), and recent works have focused on improving the motif-finding performance by integrating additional information into comparative [2] and discriminative motif discovery [3].

Despite the relative long history and the critical roles of motif discovery in bioinformatics, many issues are still unsolved and controversial. First, the target set of sequences is assumed to have only one motif, but this assumption is often incorrect. For example, a recent study examining the binding specificities of 104 mouse TFs observed that nearly half of the TFs recognize multiple sequence motifs [4]. Second, it is unclear how to select the target set on which over-represented motifs are returned. The target set of sequences is often constructed from genome-wide binding location data (ChIP-chip or ChIP-seq) or gene expression microarray data. However, there is no clear way to partition the data into target and background sets in general. Third, a unified algorithm which is applicable to diverse motif discovery problems is solely needed to provide a principled framework for developing more complex models.

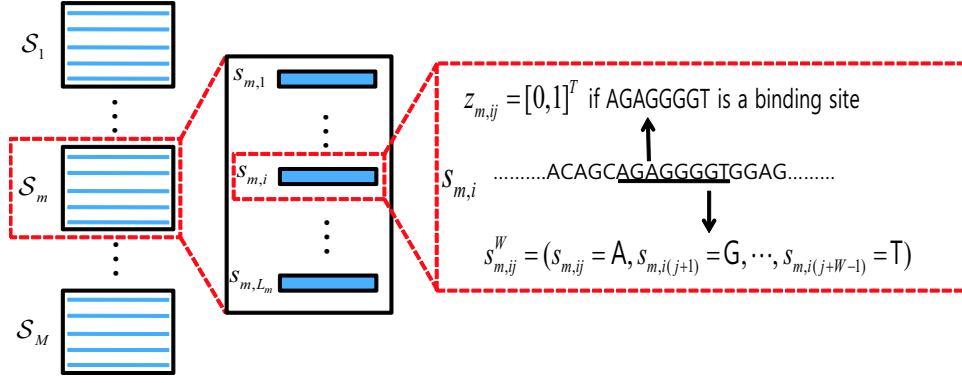

Figure 1: Notation illustration.

These considerations motivate us to develop a generative probabilistic framework for learning multiple motifs on multiple sets of sequences. One can view our framework as an extension of the classic sequence models such as the two-component mixture (TCM) [5] and the zero or one occurrence per sequence (ZOOPS) [6] models in which sequences are partitioned into two clusters, depending on whether or not they contain a motif. In this paper, we make use of a finite mixture model to partition the multiple sequence sets into clusters having distinct sequence motifs, which improves the motif-finding performance over the classic models by enhancing signal-to-noise ratio of input sequences. We also show how our algorithm can be applied into three different problems by simply changing the way of constructing multiple sets from input sequences without any algorithmic modifications.

## 2  Problem formulation

We are given $M$ sets of DNA sequences $\mathcal{S} = \{\mathcal{S}_1, \ldots, \mathcal{S}_M\}$ to be grouped according to the type of motif involved with, in which each set is associated with only a single motif but multiple binding sites are present in each sequence. A set of DNA sequences $\mathcal{S}_m = \{s_{m,1}, \ldots, s_{m,L_m}\}$ is a collection of strings $s_{m,i}$ of length $|s_{m,i}|$ over the alphabet $\Sigma = \{A, C, G, T\}$. To allow for a variable number of binding sites per sequence, we represent each sequence $s_{m,i}$ as a set of overlapping subsequences $s_{m,ij}^{W} = (s_{m,ij}, s_{m,i(j+1)}, \ldots, s_{m,i(j+W-1)})$ of length $W$ starting at position $j \in \mathcal{I}_{m,i}$, where $s_{m,ij}$ denotes the letter at position $j$ and $\mathcal{I}_{m,i} = \{1, \ldots, |s_{m,i}| - W + 1\}$, as shown in Fig. 1. We introduce a latent variable matrix $z_{m,i} \in \mathbb{R}^{2 \times |\mathcal{I}_{m,i}|}$ in which the $j^{th}$ column vector $z_{m,ij}$ is a 2-dimensional binary random vector $[z_{m,ij1}, z_{m,ij2}]^{\top}$ such that $z_{m,ij} = [0, 1]^{\top}$ if a binding site starts at position $j \in \mathcal{I}_{m,i}$, otherwise, $z_{m,ij} = [1, 0]^{\top}$. We also introduce $K$-dimensional binary random vectors $t_m \in \mathbb{R}^K$ ($t_{m,k} \in \{0, 1\}$ and $\sum_k t_{m,k} = 1$) for $m = 1, \ldots, M$, which involve partitioning the sequence sets $\mathcal{S}$ into $K$ disjoint clusters, where sets in the same cluster are associated with the same common motif.

For a motif model, we use a position-frequency matrix whose entries correspond to probability distributions (over the alphabet $\Sigma$) of each position within a binding site. We denote by $\Theta_k \in \mathbb{R}^{W \times 4}$ the $k^{th}$ motif model of length $W$ over $\Sigma$, where $\Theta_{k,w}^{\top}$ represents row $w$, each entry is non-negative, $\Theta_{k,wl} \geq 0$ for $\forall w, l$, and $\sum_{l=1}^{4} \Theta_{k,wl} = 1$ for $\forall w$. The background model $\theta_0$, which describes frequencies over the alphabet within non-binding sites, is defined by a $P^{th}$ order Markov chain (represented by a $(P+1)$-dimensional conditional probability table).

Our goal is to construct a probabilistic model for DNA motif discovery where we identify multiple motifs through searching for patterns which are shared across multiple sets of sequences. Our model infers cluster-indicating latent variables (to find a good partition of $\mathcal{S}$) and learns motifs (inferring binding site-indicating latent variables $z_{m,i}$) simultaneously, where these two tasks interact with each other.

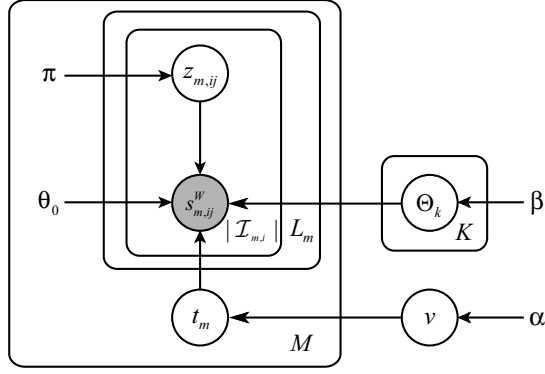

Figure 2: Graphical representation of our mixture model for $M$ sequence sets.

## 3 Mixture model for motif discovery

We assume that the distribution of $\mathcal{S}$ is modeled as a mixture of $K$ components, where it is not known in advance which mixture component underlies a particular set of sequences. We also assume that the conditional distribution of the subsequence $s_{m,ij}^W$ given $t_m$ is modeled as a mixture of two components, each of which corresponds to the motif and the background models, respectively. Then, the joint distribution of observed sequence sets $\mathcal{S}$ and (unobserved) latent variables $\mathcal{Z}$ and $\mathcal{T}$ conditioned on parameters $\Phi$ is written as:

$$p(\mathcal{S}, \mathcal{Z}, \mathcal{T}|\Phi) = \prod_m^M p(t_m|\Phi) \prod_{i=1}^{L_m} \prod_{j \in \mathcal{I}_{m,i}} p(s_{m,ij}^W|z_{m,ij}, t_m, \Phi) p(z_{m,ij}|\Phi), \tag{1}$$

where $\mathcal{Z} = \{z_{m,ij}\}$ and $\mathcal{T} = \{t_m\}$. The graphical model associated with (1) is shown in Fig. 2.

The generative process for subsequences $s_{m,ij}^W$ is described as follows. We first draw mixture weights $v = [v_1, \ldots, v_K]^\top$ (involving set clusters) from the Dirichlet distribution:

$$p(v|\alpha) \propto \prod_{k=1}^K v_k^{\frac{\alpha_k}{K} - 1}, \tag{2}$$

where $\alpha = [\alpha_1, \ldots, \alpha_K]^\top$ are the hyperparameters. Given mixture weights, we choose the cluster-indicator $t_m$ for $\mathcal{S}_m$, according to the multinomial distribution $p(t_m|v) = \prod_{k=1}^K v_k^{t_{m,k}}$. The chosen $k^{th}$ motif model $\Theta_k$ is drawn from the product of Dirichlet distributions:

$$p(\Theta_k|\beta) = \prod_{w=1}^W p(\Theta_{k,w}|\beta) \propto \prod_{w=1}^W \prod_{l=1}^4 \Theta_{k,wl}^{\beta_l - 1}, \tag{3}$$

where $\beta = [\beta_1, \ldots, \beta_4]^\top$ are the hyperparameters. The latent variables $z_{m,ij}$ indicating the starting positions of binding sites are governed by the prior distribution specified by:

$$p(z_{m,ij}|\pi) = \prod_{r=1}^2 \pi_r^{z_{m,ijr}}, \tag{4}$$

where the mixture weights $\pi = [\pi_1, \pi_2]^\top$ satisfy $\pi_1, \pi_2 \geq 0$ and $\pi_1 + \pi_2 = 1$. Finally, the subsequences $s_{m,ij}^W$ are drawn from the following conditional distribution:

$$p(s_{m,ij}^W|t_m, z_{m,ij}, \{\Theta_k\}_{k=1}^K, \theta_0) = p(s_{m,ij}^W|\theta_0)^{z_{m,ij1}} \prod_{k=1}^K (p(s_{m,ij}^W|\Theta_k)^{z_{m,ij2}})^{t_{m,k}}, \tag{5}$$

where

$$p(s_{m,ij}^W|\theta_0) = \prod_{w=1}^W \prod_{l=1}^4 \theta_{0l}^{\delta(l,s_{m,i(j+w-1)})}, \quad p(s_{m,ij}^W|\Theta_k) = \prod_{w=1}^W \prod_{l=1}^4 \Theta_{k,wl}^{\delta(l,s_{m,i(j+w-1)})},$$

where $\delta(l, s_{m,i(j+w-1)})$ is an indicator function which equals 1 if $s_{m,i(j+w-1)} = l$, and otherwise 0. Here, the background model is specified by the $0^{th}$-order Markov chain for notational simplicity.

Several assumptions simplify this generative model. First, the width $W$ of the motif model and the number $K$ of set clusters are assumed to be known and fixed. Second, the mixture weights $\pi$ together with the background model $\theta_0$ are treated as parameters to be estimated. We assume the hyperparameters $\alpha$ and $\beta$ are set to fixed and known constants. The full set of parameters and hyper-parameters will be denoted by $\Phi = \{\alpha, \beta, \pi, \theta_0\}$. Extension to double stranded DNA sequences is obvious and omitted here due to the lack of space.

Our model builds upon the existing TCM model proposed by [5] where the EM algorithm is applied to learn a motif on a single target set. This model actually generates subsequences instead of sequences themselves. An alternative model which explicitly generates sequences has been proposed based on Gibbs sampling [7, 8]. Note that our model is reduced to the TCM model if $K$, the number of set clusters, is set to one.

Our model shares some similarities with the recent Bayesian hierarchical model in [9] which also uses a mixture model to cluster discovered motifs. The main difference is that they focus on clustering motifs already discovered, and in our formulation, we try to cluster sequence sets and discover motifs simultaneously.

## 4  Inference by Gibbs sampling

We find the configurations of $\mathcal{Z}$ and $\mathcal{T}$ by maximizing the posterior distribution over latent variables:

$$\mathcal{Z}^*, \mathcal{T}^* = \arg\max_{\mathcal{Z},\mathcal{T}} p(\mathcal{Z}, \mathcal{T}|\mathcal{S}, \Phi). \tag{6}$$

To this end, we use Gibbs sampling to find the posterior modes by drawing samples repeatedly from the posterior distribution over $\mathcal{Z}$ and $\mathcal{T}$. We will derive a Gibbs sampler for our generative model in which the set mixture weights $v$ and motif models $\{\Theta_k\}_{k=1}^K$ are integrated out to improve the convergence rate and the cost per iteration [8].

The critical quantities needed to implement the Gibbs sampler are the full conditional distributions for $\mathcal{Z}$ and $\mathcal{T}$. We first derive the relevant full conditional distribution over $t_m$ conditioned on the set cluster assignments of all other sets, $\mathcal{T}_{\backslash m}$, the latent positions $\mathcal{Z}$, and the observed sets $\mathcal{S}$. By applying Bayes' rule, Fig. 2 implies that this distribution factorizes as follows:

$$p(t_{m,k} = 1|\mathcal{T}_{\backslash m}, \mathcal{S}, \mathcal{Z}, \Phi) \quad \propto \quad p(t_{m,k} = 1|\mathcal{T}_{\backslash m}, \alpha)p(\mathcal{S}_m, \mathcal{Z}_m|\mathcal{T}, \mathcal{S}_{\backslash m}, \mathcal{Z}_{\backslash m}, \Phi), \tag{7}$$

where $\mathcal{Z}_{\backslash m}$ denotes the entries of $\mathcal{Z}$ other than $\mathcal{Z}_m = \{z_{m,i}\}_{i=1}^{L_m}$, and $\mathcal{S}_{\backslash m}$ is similarly defined. The first term represents the predictive distribution of $t_m$ given the other set cluster assignments $\mathcal{T}_{\backslash m}$, and is given by marginalizing the set mixture weights $v$:

$$p(t_{m,k} = 1|\mathcal{T}_{\backslash m}, \alpha) \quad = \quad \int_v p(t_{m,k} = 1|v)p(v|\mathcal{T}_{\backslash m}, \alpha)dv = \frac{N_k^{-m} + \frac{\alpha_k}{K}}{M - 1 + \alpha_k} \tag{8}$$

where $N_k^{-m} = \sum_{n \neq m} \delta(t_{n,k}, 1)$. Note that $N_k^{-m}$ counts the number of sets currently assigned to the $k^{th}$ set cluster excluding the $m^{th}$ set. The model's Markov structure implies that the second term of (7) depends on the current assignments $\mathcal{T}_{\backslash m}$ as follows:

$$p(\mathcal{S}_m, \mathcal{Z}_m|t_{m,k} = 1, \mathcal{T}_{\backslash m}, \mathcal{S}_{\backslash m}, \mathcal{Z}_{\backslash m}, \Phi)$$

$$= \int_{\Theta_k} p(\mathcal{S}_m, \mathcal{Z}_m|t_{m,k} = 1, \Theta_k, \Phi)p(\Theta_k|\{\mathcal{S}_n, \mathcal{Z}_n|t_{n,k} = 1, n \neq m\}, \Phi)d\Theta_k$$

$$= \left[\prod_{i=1}^{L_m}\prod_{j \in \mathcal{I}_{m,i}} p(z_{m,ij}|\pi) \prod_{j \in \mathcal{I}_{m,i}; z_{m,ij2}=0} p(s_{m,ij}^W|z_{m,ij2} = 0, \theta_0)\right]$$

$$\left[\prod_{w=1}^{W}\left\{\frac{\Gamma(\sum_l(N_{wl}^{-m} + \beta_l))}{\Gamma(\sum_l(N_{wl} + \beta_l))}\frac{\prod_l \Gamma(N_{wl} + \beta_l)}{\prod_l \Gamma(N_{wl}^{-m} + \beta_l)}\right\}\right], \tag{9}$$

where $N_{wl} = N_{wl}^{-m} + N_{wl}^{m}$ and

$$N_{wl}^{-m} = \sum_{t_{n,k}=1, n \neq m} \sum_{i=1}^{L_n} \sum_{j \in \mathcal{I}_{n,i}, z_{n,ij2}=1} \delta(s_{n,i(j+w-1)}, l)$$

$$N_{wl}^{m} = \sum_{i=1}^{L_m} \sum_{j \in \mathcal{I}_{m,i}, z_{m,ij2}=1} \delta(s_{m,i(j+w-1)}, l).$$

Note that $N_{wl}^{-m}$ counts the number of letter $l$ at position $w$ within currently assigned binding sites excluding the ones of the $m^{th}$ set. Similarly, $N_{wl}^{m}$ denotes the number of letter $l$ at position $w$ within bindings sites of the $m^{th}$ set.

We next derive the full conditional distribution of $z_{m,ij}$ given the remainder of the variables. Integrating over the motif model $\Theta_k$, we then have the following factorization:

$$p(z_{m,ij} | \mathcal{Z}_{\backslash m,ij}, \mathcal{S}, t_{m,k} = 1, \mathcal{T}_{\backslash m}, \Phi) \propto \int_{\Theta_k} \prod_{t_{n,k}=1} p(\mathcal{Z}_n, \mathcal{S}_n | \Theta_k) p(\Theta_k | \beta) d\Theta_k$$

$$\propto \left[ \prod_{t_{n,k}=1} \prod_{i=1}^{L_n} \prod_{j=1}^{\mathcal{I}_{n,i}} p(z_{n,ij} | \pi) \prod_{i=1}^{L_n} \prod_{j \in \mathcal{I}_{n,i}, z_{n,ij2}=0} p(s_{n,ij}^{W} | \theta_0) \right] \left[ \prod_{w=1}^{W} \frac{\prod_l \Gamma(N_{wl} + \beta_l)}{\Gamma(\sum_l (N_{wl} + \beta_l))} \right] \quad (10)$$

where $\mathcal{Z}_{\backslash m,ij}$ denotes the entries of $\mathcal{Z}$ other than $z_{m,ij}$. For the purpose of sampling, the ratio of the posterior distribution of $z_{m,ij}$ is given by:

$$\frac{p(z_{m,ij2} = 1 | \mathcal{Z}_{\backslash m,ij}, \mathcal{S}, \mathcal{T}, \Phi)}{p(z_{m,ij2} = 0 | \mathcal{Z}_{\backslash m,ij}, \mathcal{S}, \mathcal{T}, \Phi)} = \frac{\pi_2}{\pi_1 p(s_{m,ij}^{W} | \theta_0)} \prod_{w=1}^{W} \frac{\sum_{l=1}^{4} (N_{wl}^{-m,ij} + \beta) \delta(s_{m,i(j+w-1)}, l)}{\sum_{l=1}^{4} (N_{wl}^{-m,ij} + \beta)},$$

where $N_{wl}^{-m,ij} = \sum_{t_{n,k}=1} \sum_{i=1}^{L_n} \sum_{j' \in \mathcal{I}_{n,i}, j' \neq j, z_{n,ij'2}=1} \delta(s_{n,i(j'+w-1)}, l)$. Note that $N_{wl}^{-m,ij}$ denotes the number of letter $l$ at position $w$ within currently assigned binding sites other than $z_{m,ij}$.

Combining (7) with (10) is sufficient to define the Gibbs sampler for our finite mixture model. To provide a convergence measure, we derive the following objective function based on the log of the posterior distribution:

$$\log p(\mathcal{Z}, \mathcal{T} | \mathcal{S}, \Phi) \propto \log p(\mathcal{Z}, \mathcal{T}, \mathcal{S} | \Phi)$$

$$\propto \sum_{m=1}^{M} \sum_{i=1}^{L_m} \sum_{j=1}^{\mathcal{I}_{m,i}} \log p(z_{m,ij} | \pi) + \sum_{m=1}^{M} \sum_{i=1}^{L_m} \sum_{j \in \mathcal{I}_{mi}; z_{m,ij2}=0} \log p(s_{m,ij}^{W} | \theta_0)$$

$$+ \sum_{k=1}^{K} \sum_{w=1}^{W} \left\{ \sum_l \log \Gamma(N_{wl}^{k} + \beta_l) - \log \Gamma(\sum_l (N_{wl}^{k} + \beta_l)) \right\} + \sum_{k=1}^{K} \log \Gamma(N_k + \frac{\alpha_k}{K}), (11)$$

where $N_k = \sum_m \delta(t_{m,k}, 1)$ and $N_{wl}^{k} = \sum_{t_{m,k}=1} \sum_{i=1}^{L_m} \sum_{j \in \mathcal{I}_{m,i}, z_{m,ij2}=1} \delta(s_{m,i(j+w-1)}, l)$.

## 5 Results

We evaluated our motif-finding algorithm on the three different tasks: (1) filtering out undesirable noisy sequences, (2) incorporating evolutionary conservation information, and (3) clustering DNA sequences based on the learned motifs (Fig. 3). In the all experiments, we fixed the hyper-parameters so that $\alpha_k = 1$ and $\beta_l = 0.5$.

### 5.1 Data sets and evaluation criteria

We first examined the yeast ChIP-chip data published by [10] to investigate the effect of filtering out noisy sequences from input sequences on identifying true binding sites. We compiled 156 sequence-sets by choosing TFs having consensus motifs in the literature [11]. For each sequence-set, we defined its sequences to be probe sequences that are bound with $P$-value $\leq 0.001$.

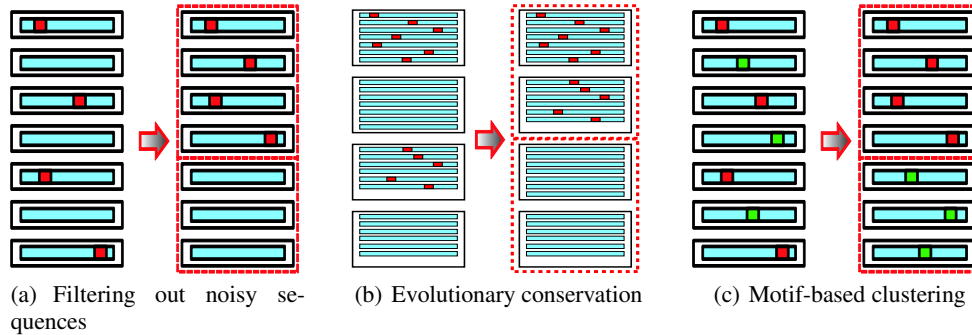

(a) Filtering out noisy sequences     (b) Evolutionary conservation     (c) Motif-based clustering

Figure 3: Three different ways of constructing multiple sequence sets. Black rectangles: sequence sets, Blue bars: sequences, Red dashed rectangles: set clusters, Red and green rectangles: motifs.

To apply our algorithm into the comparative motif discovery problem, we compiled orthologous sequences for each probe sequence of the yeast ChIP-chip data based on the multiple alignments of seven species of Saccharomyces (S. cerevisiae, S. paradoxus, S. mikatae, S. kudriavzevii, S. bayanus, S. castelli, and S. kluyveri) [12]. In the experiments using the ChIP-chip data, the motif width was set to 8 and a fifth-order Markov chain estimated from the whole yeast intergenic sequences was used to describe the background model. We fixed the mixture weights $\pi$ so that $\pi_2 = 0.001$.

We next constructed the ChIP-seq data for human neuron-restrictive silence factor (NRSF) to determine whether our algorithm can be applied to partition DNA sequences into biologically meaningful clusters [13]. The data consist of 200 sequence segments of length 100 from all peak sites with the top 10% binding intensity ($\geq 500$ ChIP-seq reads), where most sequences have canonical NRSF-binding sites. We also added 13 sequence segments extracted from peak sites ($\geq 300$ reads) known to have noncanonical NRSF-binding sites, resulting in 213 sequences. In the experiment using the ChIP-seq data, the motif width was set to 30 and a zero-order Markov chain estimated from the 213 sequence segments was used to describe the background model. We fixed the mixture weights $\pi$ so that $\pi_2 = 0.005$.

In the experiments using the yeast ChIP-chip data, we used the inter-motif distance to measure the quality of discovered motifs [10]. Specifically, an algorithm will be called *successful* on a sequence set only if at least one of the position-frequency matrices constructed from the identified binding sites is at a distance less than 0.25 from the literature consensus [14].

## 5.2   Filtering out noisy sequences

Selecting target sequences from the ChIP-chip measurements is largely left to users and this choice is often unclear. Our strategy of constructing sequence-sets based on the binding $P$-value cutoff would be exposed to danger of including many irrelevant sequences. In practice, the inclusion of noisy sequences in the target set is a serious obstacle in the success of motif discovery. One possible solution is to cluster input sequences into two smaller sets of target and noisy sequences based on sequence similarity, and predict motifs from the clustered target sequences with the improved signal-to-noise ratio. This two-step approach has been applied to only protein sequences because DNA sequences do not share much similarity for effective clustering [15].

One alternative approach is to seek a better sequence representation based on motifs. To this end, we constructed multiple sets by treating each sequence of a particular yeast ChIP-chip sequence-set as one set (Fig. 3(a)). We examined the ability of our algorithm to find a correct motif with two different numbers of clusters: $K = 1$ (without filtering) and $K = 2$ (clustering into two subsets of true and noisy sequences). We ran each experiment five times with different initializations and reported means with $\pm 1$ standard error. Figure 4 shows that the filtering approach ($K = 2$) outperforms the baseline method ($K = 1$) in general, with the increasing value of the $P$-value cutoff. Note that the ZOOPS or TCM models can also handle noisy sequences by modeling them with only a background model [5, 6]. But we allow noisy sequences to have a *decoy motif* (randomly occurring sequence

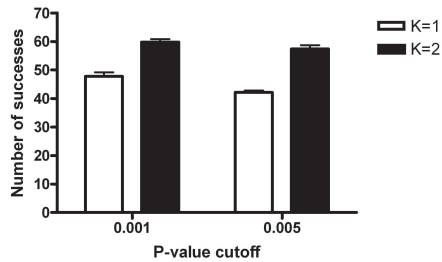

Figure 4: Effect of filtering out noisy sequences on the number of successfully identified motifs on the yeast ChIP-chip data. $K = 1$: without filtering, $K = 2$: clustering into two subsets.

patterns or repeating elements) which is modeled with a motif model. Because our model can be reduced to these classic models by setting $K = 1$, we concluded that noisy sequences were better represented by our clustering approach than the previous ones using the background model (Fig. 4).

Two additional lines of evidence indicated that our filtering approach enhances the signal-to-noise ratio of the target set. First, we compared the results of our filtering approach with that of other baseline methods (AlignAce [16], MEME [6], MDScan [17], and PRIORITY-$\mathcal{U}$ [11]) on the same yeast ChIP-chip data. For AlignAce, MEME and MDScan, we used the results reported by [14]; for PRIORITY-$\mathcal{U}$, we used two different results reported by [14, 11] according to different sampling strategy. We expected that our model would perform better than these four methods because they try to remove noisy sequences based on the classic models. By comparing the results of Fig. 4 and Table 1, we see that our algorithm still performs better. Second, we also compared our model with DRIM specifically designed to dynamically select the target set from the list of sorted sequences according to the binding $P$-values of ChIP-chip measurements. For DRIM, we used the result reported by [18]. Because DRIM does not produce any motifs when they are not statistically enriched at the top of the ranked list, we counted the number of successfully identified motifs on the sequence-sets where DRIM generated significant motifs. Our method (number of successes is 16) was slightly better than DRIM (number of successes is 15).

### 5.3 Detecting evolutionary conserved motifs

Comparative approach using evolutionary conservation information has been widely used to improve the performance of motif-finding algorithms because functional TF binding sites are likely to be conserved in orthologous sequences. To incorporate conservation information into our clustering framework, orthologous sequences of each sequence of a particular yeast ChIP-chip sequence-set were considered as one set and the number of clusters was set to 2 (Fig. 3(b)). The constructed sets contain at most 7 sequences because we only used seven species of Saccharomyces. We used the single result with the highest objective function value of (11) among five runs and compared it with the results of five conservation-based motif finding algorithms on the same data set: MEME_c [10], PhyloCon [19], PhyMe [20], PhyloGibbs [21], PRIORITY-$\mathcal{C}$ [11]. For the five methods, we used the results reported by [11]. We did not compare with discriminative methods which are known to perform better at this data set because our model does not use negative sequences. Table 1 presents the motif-finding performance in terms of the number of correctly identified motifs for each algorithm. We see that our algorithm greatly outperforms the four alignment-based methods which rely on multiple or pair-wise alignments of orthologous sequences to search for motifs that are conserved across the aligned blocks of orthologous sequences. In our opinion, it is because diverged regions other than the short conserved binding sites may prevent a correct alignment. Moreover, our algorithm performs somewhat better than PRIORITY-$\mathcal{C}$, which is a recent alignment-free method. We believe that it is because the signal-to-noise ratio of the input target set is enhanced by clustering.

### 5.4 Clustering DNA sequences based on motifs

To examine the ability of our algorithm to partition DNA sequences into biologically meaningful clusters, we applied our algorithm to the NRSF ChIP-seq data which are assumed to have two

Table 1: Comparison of the number of successfully identified motifs on the yeast ChIP-chip data for different methods. NC: Non-conservation, EC: Evolutionary conservation, A: Alignment-based, AF: Alignment-free, C: Clustering.

| Method | Description | # of successes |
|---|---|---|
| AlignAce | NC | 16 |
| MEME | NC | 35 |
| MDScan | NC | 54 |
| PRIORITY-$\mathcal{U}$ | NC | 46-58 |
| MEME_c | EC + A | 49 |
| PhyloCon | EC + A | 19 |
| PhyME | EC + A | 21 |
| PhyloGibbs | EC + A | 54 |
| PRIORITY-$\mathcal{C}$ | EC + AF | 69 |
| This work | EC + AF + C | 75 |

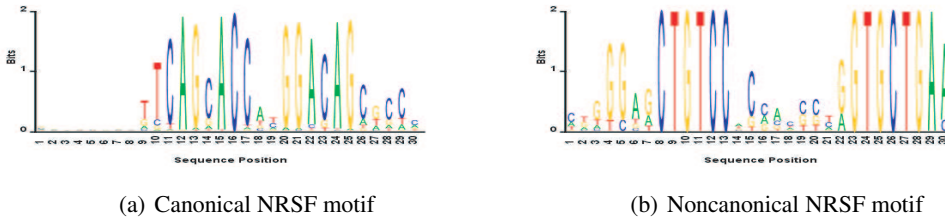

(a) Canonical NRSF motif          (b) Noncanonical NRSF motif

Figure 5: Sequence logo of discovered NRSF motifs.

different NRSF motifs (Fig. 3(c)). In this experiment, we have already known the number of clusters ($K = 2$). We ran our algorithm five times with different initializations and reported the one with the highest objective function value. Position-frequency matrices of two clusters are shown in Fig. 5. The two motifs correspond directly to the previously known motifs (canonical and non-canonical NRSF motifs). However, other motif-finding algorithms such as MEME could not return the non-canonical motif enriched in a very small set of sequences. These observations suggest that our motif-driven clustering approach is effective at inferring latent clusters of DNA sequences and can be used to find unexpected novel motifs.

## 6   Conclusions

In this paper, we have presented a generative probabilistic framework for DNA motif discovery using multiple sets of sequences where we cluster DNA sequences and learn motifs interactively. We have presented a finite mixture model with two different types of latent variables, in which one is associated with cluster-indicators and the other corresponds to motifs (transcription factor binding sites). These two types of latent variables are inferred alternatively using multiple sets of sequences. Our empirical results show that the proposed method can be applied to various motif discovery problems, depending on how to construct the multiple sets. In the future, we will explore several other extensions. For example, it would be interesting to examine the possibility of learning the number of clusters from data based on Dirichlet process mixture models, or to extend our probabilistic framework for discriminative motif discovery.

**Acknowledgments:** We thank Raluca Gordân for providing the literature consensus motifs and the script to compute the inter-motif distance. This work was supported by National Core Research Center for Systems Bio-Dynamics funded by Korea NRF (Project No. 2009-0091509) and WCU Program (Project No. R31-2008-000-10100-0). JKK was supported by a Microsoft Research Asia fellowship.

# References

[1] G. D. Stormo. DNA binding sites: representation and discovery. *Bioinformatics*, 16:16–23, 2000.

[2] W. W. Wasserman and A. Sandelin. Applied bioinformatics for the identification of regulatory elements. *Nature Review Genetics*, 5:276–287, 2004.

[3] E. Segal, Y. Barash, I. Simon, N. Friedman, and D. Koller. From promoter sequence to expression: a probabilistic framework. In *Proceedings of the International Conference on Research in Computational Molecular Biology*, pages 263–272, 2002.

[4] G. Badis, M. F. Berger, A. A. Philippakis, S. Talukder, A. R. Gehrke, S. A. Jaeger, E. T. Chan, G. Metzler, A. Vedenko, X. Chen, H. Kuznetsov, C. F. Wang, D. Coburn, D. E. Newburger, Q. Morris, T. R. Hughes, and M. L. Bulyk. Diversity and complexity in DNA recognition by transcription factors. *Science*, 324:1720–1723, 2009.

[5] T. L. Bailey and C. Elkan. Fitting a mixture model by expectation maximization to discover motifs in biopolymers. In *Proceedings of the International Conference Intelligent Systems for Molecular Biology*, 1994.

[6] T. L. Bailey and C. Elkan. The value of prior knowledge in discovering motifs with MEME. In *Proceedings of the International Conference Intelligent Systems for Molecular Biology*, 1995.

[7] C. E. Lawrence, S. F. Altschul, M. S. Boguski, J. S. Liu, A. F. Neuwald, and J. C. Wootton. Detecting subtle sequence signals: a Gibbs sampling strategy for multiple alignment. *Science*, 262:208–214, 1993.

[8] J. S. Liu, A. F. Neuwald, and C. E. Lawrence. Bayesian models for multiple local sequence alignment and Gibbs sampling strategies. *Journal of the American Statistical Association*, 90:1156–1170, 1995.

[9] S. T. Jensen and J. S. Liu. Bayesian clustering of transcription factor binding motifs. *Journal of the American Statistical Association*, 103:188–200, 2008.

[10] C. T. Harbison, D. B. Gordon, T. I. Lee, N. J. Rinaldi, K. D. Macisaac, T. W. Danford, N. M. Hannett, J. B. Tagne, D. B. Reynolds, J. Yoo, E. G. Jennings, J. Zeitlinger, D. K. Pokholok, M. Kellis, P. A. Rolfe, K. T. Takusagawa, E. S. Lander, D. K. Gifford, E. Fraenkel, and R. A. Young. Transcriptional regulatory code of a eukaryotic genome. *Nature*, 431:99–104, 2004.

[11] R. Gordan, L. Narlikar, and A. J. Hartemink. A fast, alignment-free, conservation-based method for transcription factor binding site discovery. In *Proceedings of the International Conference on Research in Computational Molecular Biology*, pages 98–111, 2008.

[12] A. Siepel, G. Bejerano, J. S. Pedersen, A. S. Hinrichs, M. Hou, K. Rosenbloom, H. Clawson, J. Spieth, L. W. Hillier, S. Richards, G. M. Weinstock, R. K. Wilson, R. A. Gibbs, W. J. Kent, W. Miller, and D. Haussler. Evolutionarily conserved elements in vertebrate, insect, worm, and yeast genomes. *Genome Research*, 15:1034–1050, 2005.

[13] D. S. Johnson, A. Mortazavi, R. M. Myers, and B. Wold. Genome-wide mapping of in vivo protein-DNA interactions. *Science*, 316:1497–1502, 2007.

[14] L. Narlikar, R. Gordan, and A. J. Hartemink. Nucleosome occupancy information improves de novo motif discovery. In *Proceedings of the International Conference on Research in Computational Molecular Biology*, pages 107–121, 2007.

[15] S. Kim, Z. Wang, and M. Dalkilic. igibbs: improving gibbs motif sampler for proteins by sequence clustering and iterative pattern sampling. *Proteins*, 66:671–681, 2007.

[16] F. P. Roth, J. D. Hughes, P. W. Estep, and G. M. Church. Finding DNA regulatory motifs within unaligned noncoding sequences clustered by whole-genome mRNA quantitation. *Nature Biotechnology*, 16:939–945, 1998.

[17] X. S. Liu, D. L. Brutlag, and J. S. Liu. An algorithm for finding protein-DNA binding sites with applications to chromatin-immunoprecipitation microarray experiments. *Nature Biotechnology*, 20:835–839, 2002.

[18] E. Eden, D. Lipson, S. Yogev, and Z. Yakhini. Discovering motifs in ranked lists of DNA sequences. *PLoS Computational Biology*, 3:e39, 2007.

[19] T. Wang and G. D. Stormo. Combining phylogenetic data with co-regulated genes to identify regulatory motifs. *Bioinformatics*, 19:2369–2380, 2003.

[20] S. Sinha, M. Blanchette, and M. Tompa. PhyME: a probabilistic algorithm for finding motifs in sets of orthologous sequences. *BMC Bioinformatics*, 5:170, 2004.

[21] R. Siddharthan, E. D. Siggia, and E. van Nimwegen. PhyloGibbs: a gibbs sampling motif finder that incorporates phylogeny. *PLoS Computational Biology*, 1:e67, 2005.

